# Unsupervised Classifiers, Mutual Information and 'Phantom Targets'

John S. Bridle
Anthony J.R. Heading
Defence Research Agency
St. Andrew's Road, Malvern
Worcs. WR14 3PS, U.K.

David J.C. MacKay
California Institute of Technology 139-74
Pasadena CA 91125 U.S.A

## Abstract

We derive criteria for training adaptive classifier networks to perform unsupervised data analysis. The first criterion turns a simple Gaussian classifier into a simple Gaussian mixture analyser. The second criterion, which is much more generally applicable, is based on mutual information. It simplifies to an intuitively reasonable difference between two entropy functions, one encouraging 'decisiveness,' the other 'fairness' to the alternative interpretations of the input. This 'firm but fair' criterion can be applied to any network that produces probability-type outputs, but it does not necessarily lead to useful behavior.

## 1 Unsupervised Classification

One of the main distinctions made in discussing neural network architectures, and pattern analysis algorithms generally, is between supervised and unsupervised data analysis. We should therefore be interested in any method of building bridges between techniques in these two categories. For instance, it is possible to use an unsupervised system such as a Boltzmann machine to learn the joint distribution of inputs and a teacher's classification labels. The particular type of bridge we seek is a method of taking a supervised pattern classifier and turning it into an unsupervised data analyser. That is, we are interested in methods of "bootstrapping" classifiers.

Consider a classifier system. Its input is a vector $\mathbf{x}$, and the output is a probability vector $\mathbf{y}(\mathbf{x})$. (That is, the elements of $\mathbf{y}$ are positive and sum to 1.) The elements of $\mathbf{y}$, $(y_i(\mathbf{x}), i = 1 \ldots N_c)$ are to be taken as the probabilities that $\mathbf{x}$ should be assigned to each of $N_c$ classes. (Note that our definition of classifier does not include a decision process.)

To enforce the conditions we require for the output values, we recommend using a generalised logistic (normalised exponential, or SoftMax) output stage. We call the unnormalised log probabilities of the classes $a_i$, and the softmax performs:

$$y_i = e^{a_i}/Z \quad \text{with} \quad Z = \sum_i e^{a_i} \tag{1}$$

Normally the parameters of such a system would be adjusted using a training set comprising examples of inputs and corresponding classes, $\{(\mathbf{x}_i, c_i)\}$. We assume that the system includes means to convert derivatives of a training criterion with respect to the outputs into a form suitable for adjusting the values of the parameters, for instance by "backpropagation".

Imagine however that we have *unlabelled* data, $\mathbf{x}_m, m = 1 \ldots N_{ts}$, and wish to use it to 'improve' the classifier. We could think of this as self–supervised learning, to hone an already good system on lots of easily–obtained unlabelled real–world data, or to adapt to a slowly changing environment, or as a way of turning a classifier into some sort of cluster analyser. (Just what kind depends on details of the classifier itself.) The ideal method would be theoretically well-founded, general-purpose (independent of the details of the classifier), and computationally tractable.

One well known approach to unsupervised data analysis is to minimise a reconstruction error: for linear projections and squared euclidean distance this leads to principal components analysis, while reference-point based classifiers lead to vector quantizer design methods, such as the LBG algorithm . Variants on VQ, such as Kohonen's feature maps, can be motivated by requiring robustness to distortions in the code space . Reconstruction error is only available as a training criterion if reconstruction is defined: in general we are only given class label probabilities.

## 2 A Data Likelihood Criterion

For the special case of a Gaussian clustering of an unlabelled data set, it was demonstrated in [1] that gradient ascent on the likelihood of the data has an appealing interpretation in terms of backpropagation in an equivalent unit-Gaussian classifier network: for each input $\mathbf{x}$ presented to the network, the output $\mathbf{y}$ is doubled to give 'phantom targets' $\mathbf{t} = 2\mathbf{y}$; when the derivatives of the log likelihood criterion $J = -\Sigma_i t_i \log y_i$ relative to these targets are propagated back through the network, it turns out that the resulting gradient is identical to the gradient of the likelihood of the data given a Gaussian mixture model.

For the unit-Gaussian classifier, the activations $a_i$ in (1) are

$$a_i = -|\mathbf{x} - \mathbf{w}_i|^2, \tag{2}$$

so the outputs of the network are

$$y_i = P(\text{class} = i \mid \mathbf{x}, \mathbf{w}) \tag{3}$$

where we assume the inputs are drawn from equi-probable unit-Gaussian distributions with the mean of the distribution of the $i^{\text{th}}$ class equal to $\mathbf{w}_i$.

This result was only derived in a limited context, and it was speculated that it might be generalisable to arbitrary classification models. The above phantom target rule

has been re-derived for a larger class of networks [4], but the conditions for strict applicability are quite severe. Briefly, there should be exponential density functions for each class, and the normalizing factors for these densities should be independent of the parameters. Thus Gaussians with fixed covariance matrices are acceptable, but variable covariances are not, and neither are linear transformations preceeding the Gaussians.

The next section introduces a new objective function which is independent of details of the classifier.

## 3    Mutual Information Criterion

Intuitively, an unsupervised adaptive classifier is doing a plausible job if its outputs usually give a fairly clear indication of the class of an input vector, and if there is also an even distribution of input patterns between the classes. We could label these desiderata 'decisive' and 'fair' respectively. Note that it is trivial to achieve either of them alone. For a poorly regularised model it may also be trivial to achieve both.

There are several ways to proceed. We could devise *ad–hoc* measures corresponding to our notions of decisiveness and fairness, or we could consider particular types of classifier and their unsupervised equivalents, seeking a general way of turning one into the other. Our approach is to return to the general idea that the class predictions should retain as much information about the input values as possible. We use a measure of the information about $\mathbf{x}$ which is conveyed by the output distribution, *i.e.* the mutual information between the inputs and the outputs. We interpret the outputs $\mathbf{y}$ as a probability distribution over a discrete random variable $c$ (the class label), thus $\mathbf{y} = p(c|\mathbf{x})$. The mutual information between $\mathbf{x}$ and $c$ is

$$\mathcal{I}(c\,;\mathbf{x}) \;=\; \iint dc\,d\mathbf{x}\,p(c,\mathbf{x})\log\frac{p(c,\mathbf{x})}{p(c)p(\mathbf{x})} \tag{4}$$

$$=\; \int d\mathbf{x}\,p(\mathbf{x})\int dc\,p(c|\mathbf{x})\log\frac{p(c|\mathbf{x})}{p(c)} \tag{5}$$

$$=\; \int d\mathbf{x}\,p(\mathbf{x})\int dc\,p(c|\mathbf{x})\log\frac{p(c|\mathbf{x})}{\int d\mathbf{x}\,p(\mathbf{x})p(c|\mathbf{x})} \tag{6}$$

The elements of this expression are separately recognizable:

$\int d\mathbf{x}\,p(\mathbf{x})(\cdot)$ is equivalent to an average over a training set $\frac{1}{N_{ts}}\sum_{ts}(\cdot)$;

$p(c|\mathbf{x})$ is simply the network output $y_c$;

$\int dc\,(\cdot)$ is a sum over the class labels and corresponding network outputs.

Hence:

$$\mathcal{I}(c\,;\mathbf{x}) \;=\; \frac{1}{N_{ts}}\sum_{ts}\sum_{i=1}^{N_c} y_i \log\frac{y_i}{\overline{y}_i} \tag{7}$$

$$= -\sum_{i=1}^{N_c} \overline{y}_i \log \overline{y}_i + \frac{1}{N_{ts}} \sum_{ts} \sum_{i=1}^{N_c} y_i \log y_i \tag{8}$$

$$= \mathcal{H}(\overline{\mathbf{y}}) - \overline{\mathcal{H}(\mathbf{y})} \tag{9}$$

The objective function $I$ is the difference between the entropy of the average of the outputs, and the average of the entropy of the outputs, where both averages are over the training set. $\mathcal{H}(\overline{\mathbf{y}})$ has its maximum value when the average activities of the separate outputs are equal – this is 'fairness'. $\overline{\mathcal{H}(\mathbf{y})}$ has its minimum value when one output is full on and the rest are off for every training case – this is 'firmness'.

We now evaluate $I$ for the training set. and take the gradient of $I$.

## 4   Gradient descent

To use this criterion with back–propagation network training, we need its derivatives with respect to the network outputs.

$$\frac{\partial I(c\,;\mathbf{x})}{\partial y_i} = \frac{\partial}{\partial y_i} \frac{1}{N_{ts}} \sum_{ts} \sum_{i=1}^{N_c} y_i \log \frac{y_i}{\overline{y}_i} \tag{10}$$

$$= \frac{1}{N_{ts}} \sum_{ts} [1 + \log y_i - 1 - \log \overline{y}_i] \tag{11}$$

$$= \frac{1}{N_{ts}} \sum_{ts} \log \frac{y_i}{\overline{y}_i} \tag{12}$$

The resulting expression is quite simple, but note that the presence of a $\overline{y}_i$ term means that two passes through the training set are required: the first to calculate the average output node activations, and the second to back–propagate the derivatives.

## 5   Illustrations

Figures 1 shows $I$ (divided by its maximum possible value, $\log N_c$) for a run of a particular unit-Gaussian classifier network. The 30 data points are drawn from a 2-d isotropic Gaussian. Figure 2 shows the fairness and firmness criteria separately. (The upper curve is 'fairness' $\mathcal{H}(\overline{\mathbf{y}})/\log N_c$, and the lower curve is 'firmness' $(1 - \overline{\mathcal{H}(\mathbf{y})}/\log N_c)$.)

The ten reference points had starting values drawn from the same distribution as the data. Figure 3 shows their movement during training. From initial positions within the data cluster, they move outwards into a circle around the data. The resulting classification regions are shown in Figure 4. (The grey level is proportional to the value of the maximum response at each point, and since the outputs are positive normalised this value drops to 0.5 or less at the decision boundaries.) We observe that the space is being partitioned into regions with roughly equal numbers of points. It might be surprising at first that the reference points do not end up near

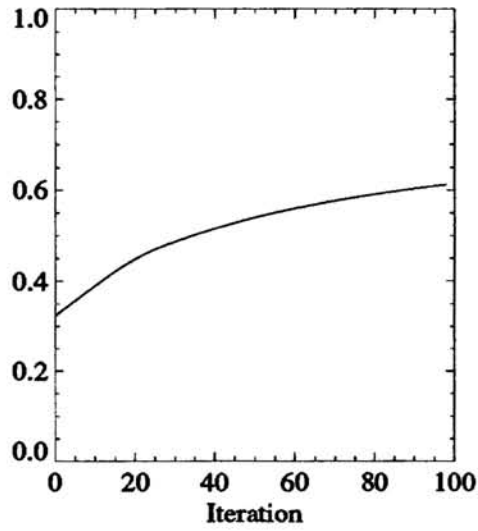

1. The M.I. criterion

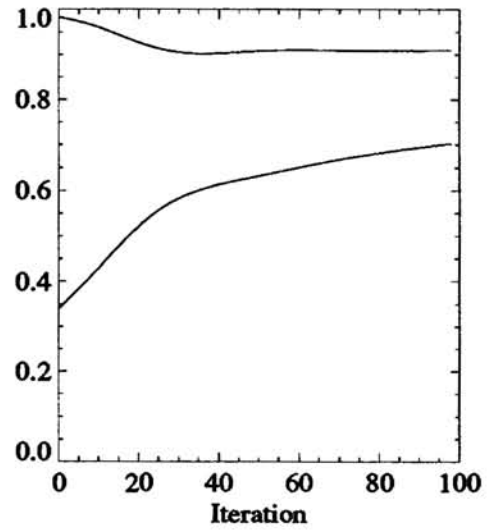

2. Firm and Fair separately

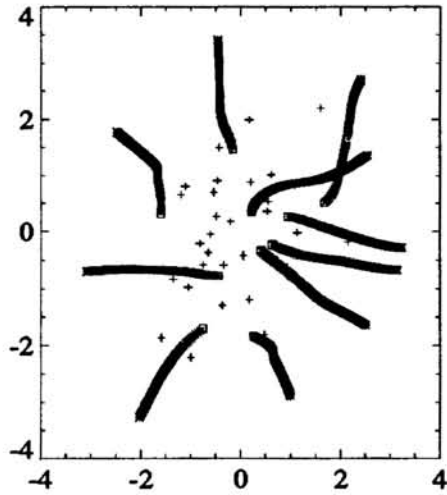

3. Tracks of reference points

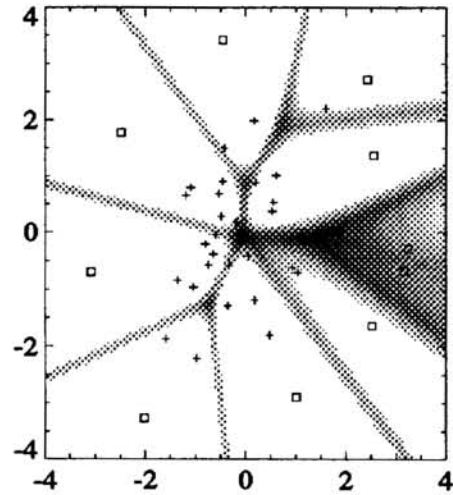

4. Decision Regions

the data. However, it is only the transformation from data $\mathbf{x}$ to outputs $\mathbf{y}$ that is being trained, and the reference points are just parameters of that transformation. As the reference points move further away from one another the decision boundaries grow firmer. In this example the fairness criterion happens to decrease in favour of the firmness, and this usually happens. We could consider different weightings of the two components of the criterion.

# 6   Comments

The usefulness of this objective function will prove will depend very much on the form of classifier that it is applied to. For a poorly regularised classifier, maximisation of the criterion alone will not necessarily lead to good solutions to unsupervised classification; it could be maximised by any implausible classification of the input that is completely hard (*i.e.* the output vector always has one 1 and all the other outputs 0), and that chops the training set into regions containing similar numbers of training points; such a solution would be one of many global maxima, regardless of whether it chopped the data into natural classes.

The meaning of a 'natural' partition in this context is, of course, rather ill-defined. Simple models often do not have the capacity to break a pattern space into highly contorted regions – the decision boundaries shown in the figure below is an example of model producing a reasonable result as a consequence of its inherent simplicity. When we use more complex models, however, we must ensure that we find simpler solutions in preference to more complex ones. Thus this criterion encourages us to pursue objective techniques for regularising classification networks [2, 3]; such techniques are probably long overdue.

# References

[1] J.S. Bridle (1988). The phantom target cluster network: a peculiar relative of (unsupervised) maximum likelihood stochastic modelling and (supervised) error backpropagation,  RSRE Research Note SP4: 66, DRA Malvern UK.

[2] D.J.C. MacKay (1991). Bayesian interpolation, submitted to *Neural computation*.

[3] D.J.C. MacKay (1991). A practical Bayesian framework for backprop networks, submitted to *Neural computation*.

[4] J S Bridle and S J Cox. Recnorm: Simultaneous normalisation and classification applied to speech recognition. In *Advances in Neural Information Processing Systems 3*. Morgan Kaufmann, 1991.

[5] J S Bridle. Training stochastic model recognition algorithms as networks can lead to maximum mutual information estimation of parameters. In *Advances in Neural Information Processing Systems 2*. Morgan Kaufmann, 1990.
